# Entangled Monte Carlo

**Seong-Hwan Jun**    **Liangliang Wang**    **Alexandre Bouchard-Côté**
Department of Statistics
University of British Columbia
{seong.jun, l.wang, bouchard}@stat.ubc.ca

## Abstract

We propose a novel method for scalable parallelization of SMC algorithms, Entangled Monte Carlo simulation (EMC). EMC avoids the transmission of particles between nodes, and instead reconstructs them from the particle genealogy. In particular, we show that we can reduce the communication to the particle weights for each machine while efficiently maintaining implicit global coherence of the parallel simulation. We explain methods to efficiently maintain a genealogy of particles from which any particle can be reconstructed. We demonstrate using examples from Bayesian phylogenetic that the computational gain from parallelization using EMC significantly outweighs the cost of particle reconstruction. The timing experiments show that reconstruction of particles is indeed much more efficient as compared to transmission of particles.

## 1   Introduction

In this paper, we focus on scalable parallelization of Monte Carlo simulation, a problem motivated by the increasingly large inference problems occurring in a variety of fields in science and engineering. Specifically, we assume that we are given a large scale inference problem involving an intractable posterior expectation, for example a Bayes estimator, and that Monte Carlo simulation is to be used to approximate the targeted expectation.

We are specifically interested in parallel Monte Carlo algorithms that scale not only in scientific-computing clusters, where node communication is fast and cheap, but also in situations where communication between nodes is limited by a combination of latency, throughput, and cost. For example, severe communication constraints arise in peer-to-peer distributed computing projects such as BOINC [1], and more generally in clusters assembled from commodity hardware.

Sequential Monte Carlo (SMC) is generally viewed as the leading candidate for massively parallel simulation, but because of particle resampling, existing implementations require the network transfer of a large number of particles and a central server with a global view on the weights carried by the particles. As a consequence, the naive communication cost grows with the size of the inference problem.

Our main contribution is a method, Entangled Monte Carlo simulation (EMC), for carrying out SMC simulation in a cluster with a communication cost per particle independent of the problem size. Our approach is fully generic and does not assume any known structure on the target distribution or the proposal used in the simulation. These desirable characteristics are achieved by limiting the contents of inter-node transmission to summary statistics on the particle weights. These summary statistics are compact and of size independent in the size of the state space of the target integral. We show that our summary statistics are sufficient, in the sense that they can be used in combination with the particle genealogy to quickly reconstruct any particle in any node of the cluster.

We will illustrate the advantage of particle reconstruction versus network transmission in the context of phylogenetic inference, a well known example where Monte Carlo simulation is both important

and challenging. In the case of the SMC sampler from [2], the cost of transmitting one particle is proportional to the product of the number of species under study, times the number of sites in the sequences, times the number of characters possible at each site.

We also introduce the algorithms needed to do these reconstructions efficiently while maintaining a distributed representation of the particle genealogies. The main algorithm is based on an alternative representation of simulation borrowed from the field of perfect simulation [3]. We demonstrate that using our algorithms, the computational cost involved in these reconstructions is negligible compared to the corresponding gains obtained from parallelization. While we describe EMC in the context of SMC simulation, it can accommodate any MCMC proposal. This is done by using the construction of artificial backward kernels [4, 5].

There is a large literature on parallelization of both MCMC and SMC algorithms. For SMC, most of the work has been on parallelization of the proposal steps [6], which is sufficient in setups such as GPU parallelization where communication between computing units is fast and cheap. However in generic clusters or peer-to-peer architectures, we argue that our more efficient parallelization of the resampling step is advantageous.

For MCMC, there is a large amount of literature on parallelization involving kernels that take the form of local Gibbs update in a graphical model. These methods allow for several blocks of variables to be updated in parallel. However, the communication cost can be high in a dense graphical model as state information needs to be synchronized. Moreover, the method is restricted to certain kinds of Gibbs kernel [7, 8, 9].

Another popular MCMC parallelization method is parallel tempering [10], where auxiliary chains are added to enable faster exploration of the space by swapping states in different chains. While parallel tempering has a low communication cost independent of the inference problem size, the additional gain of parallelism can quickly decrease as more chains are added because many swaps are needed to get from the most heated chain to the main chain.

## 2 Background

We will denote the target distribution by $\pi$, which in a Bayesian problem would correspond to a posterior distribution. The main goal is to compute the integral under $\pi$ of one or more test functions $h$, which we denote by $\pi(h)$ for short. In a Bayesian problem, this arises as the posterior expectation needed when computing a Bayes estimator. We will denote the state space by $\mathcal{S}$, i.e. $h : \mathcal{S} \to \mathbb{R}$, $\pi : \mathcal{F}_{\mathcal{S}} \to [0,1]$, where $(\mathcal{S}, \mathcal{F}_{\mathcal{S}})$ is a probability space.

### 2.1 Stochastic maps

An important concept used in the construction of our algorithms is the idea of a *stochastic map*. We start by reviewing stochastic maps in the context of a Markov chain, where it was first introduced to design perfect simulation algorithms.

Let $T : \mathcal{S} \times \mathcal{F}_{\mathcal{S}} \to [0,1]$ denote the transition kernel of a Markov chain (generally constructed by first proposing and then deciding whether to move or not using a Metropolis-Hastings (MH) ratio). A stochastic map is an equivalent view of this chain, pushing the randomness into a list of random transition functions. Formally, it is a $(\mathcal{S} \to \mathcal{S})$-valued random variable $F$ such that $T(s, A) = \mathbb{P}(F(s) \in A)$ for all state $s \in \mathcal{S}$ and event $A \in \mathcal{F}_{\mathcal{S}}$. Concretely, these maps are constructed by using the observation that $T$ is typically defined as a transformation $t(u, s)$ with $u \in [0,1]$. The most fundamental example is the case where $t$ is based on the inverse cumulative distribution method. We can then write $F(s) = t(U, s)$ for a uniform random variable $U$ on $[0,1]$.

With this notation, we get a non-standard, but useful way of formulating MCMC algorithms. First, sample $N$ stochastic maps $F_1, F_2, \ldots, F_N$, independently and identically. Second, to compute the state of the chain after $n$ transitions, simply return $F_1(F_2(\ldots(F_n(x_0))\ldots)) = F_1 \circ \cdots \circ F_n(x_0)$, for an arbitrary start state $x_0 \in \mathcal{S}$, $n \in \{1, 2, \ldots, N\}$. This representation decouples the dependencies induced by random number generation from the dependencies induced by operations on the state space. In MCMC, the latter are still not readily amenable to parallelization, and this is the motivation for using SMC as the foundation of our method. We will show in Section 3 that SMC algorithms can also be rewritten using stochastic maps.

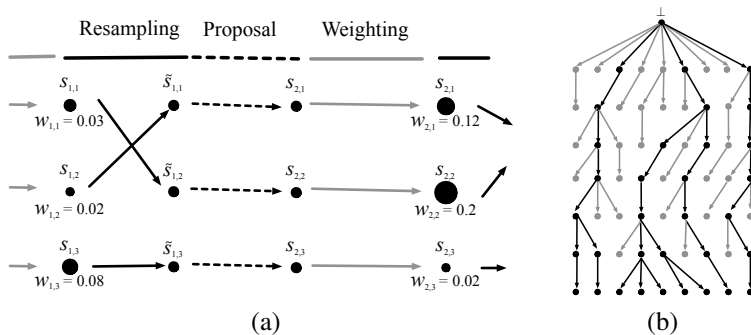

Figure 1: (a) a graphical illustration of the SMC algorithm. (b) Particle genealogy.

## 2.2 SMC algorithms

Before going over our parallel version of SMC and to keep the exposition self-contained, we review here the notation and description of standard, serial SMC algorithms from [11], which in turn is based on the SMC framework of [12, 4, 5]. The samplers used in this paper are defined using a proposal $\nu : \mathcal{S} \times \mathcal{F}_{\mathcal{S}} \to [0, 1]$. Here, $\mathcal{S}$ can be an enlarged version of the target space, with intermediate states added to ease sampling. We assume that $\pi$ has been correspondingly enlarged. The technical conditions on $\nu$ and $\pi$ are explained in [11], but for the purpose of understanding our method, only the algorithmic description of SMC given below is necessary.

SMC proceeds in a sequence of generations indexed by $r$. At each generation, the algorithm keeps in memory a weighted list of $K$ particles, $s_{r,1}, \ldots, s_{r,K} \in \mathcal{S}$, with corresponding weights $w_{r,1}, \ldots, w_{r,K}$ (see Figure 1 (a)). The weighted particles induce a distribution on $\mathcal{S}$ defined by:

$$\pi_{r,K}(A) \propto \sum_{k=1}^{K} w_{r,k} \delta_{s_{r,k}}(A), \tag{1}$$

where $A \in \mathcal{F}_{\mathcal{S}}$ is an event, and $\delta_x(A) = 1$ if $x \in A$ and 0 otherwise. We define the algorithm recursively on the generation $r$. In the base case, we set $w_{0,k} = 1/K$ for all $k \in \{1, \ldots, K\}$, and the $s_{0,k}$ are initialized to a designated start state $\perp$. Given the list of particles and weights from the previous generation $r - 1$, a new list is created in three steps. The first step can be understood as a method for pruning unpromising particles. This is done by sampling independently $K$ times from the weighted particles distribution $\pi_{r,K}$ defined above. The result of this step is that some of the particles (mostly those of low weight) will be pruned. We denote the sampled particles by $\tilde{s}_{r-1,1}, \ldots, \tilde{s}_{r-1,K}$. The second step is to create new particles, $s_{r,1}, \ldots, s_{r,K}$, by extending the partial states of each of the sampled particles from the previous iteration. This is done by sampling $K$ times from the proposal distribution, $s_{r,k} \sim \nu_{\tilde{s}_{r-1,k}}$. The third step is to compute weights for the new particles: $w_{r,k} = \alpha(\tilde{s}_{r-1,k}, s_{r,k})$, where the weight update function $\alpha$ is an easy to evaluate deterministic function $\alpha : \mathcal{S}^2 \to [0, \infty)$. We give examples in Section 4.1.

Finally, the target integral $\pi(h)$ is approximated using the weighted distribution of the last generation $R$, $\pi_{R,K}(h)$. Note that using recent work on SMC, it is possible to convert any MCMC proposals targeting a state space $\mathcal{X}$ into a valid SMC algorithm [4, 5]. This can be done for example by using an expended state space $\mathcal{S} = \mathcal{X}^R$ and by constructing an auxiliary distribution on this new space. See [4, 5] for details.

## 3  Entangled Monte Carlo Simulation

To parallelize SMC, we will view the applications of SMC proposals as a collection of stochastic maps to be shared across machines. Note that there are $K \cdot R$ proposal applications in total, which we will index by $\mathcal{I} \ni i = i(r, k) = (r(i), k(i))$ for convenience. Applying these stochastic maps, denoted by $\mathscr{F} = \{F_i : i \in \mathcal{I}\}$, is often computationally intensive (for example because of Rao-Blackwellization), and it is common to view this step as the computational bottleneck. At iteration $r$, each machine, with index $m \in \{1, \ldots, M\}$, will therefore be responsible of computing proposals

**Algorithm 1** : **EMC**$(\alpha, \nu, h, \mathcal{I}_0)$

1: $(\mathscr{F}, \mathscr{G}, \mathscr{H}) \leftarrow$ **entangle**$(\nu)$ {Section 3.3}
2: $s \leftarrow$ **empty-hashtable**
3: $\rho \leftarrow$ **empty-genealogy**
4: **init**$(s, w)$
5: **for** $r \in \{1, \ldots, R\}$ **do**
6:     **exchange**$(w_{r-1})$
7:     **resample**$(w_{r-1}, \rho, \mathcal{I}_{r-1}, \mathscr{G})$
     {Supplementary Material}
8:     $\mathcal{I}_r \leftarrow$ **allocate**$(\rho, \mathcal{I}_{r-1}, \mathscr{H})$ {Section 3.1}
9:     **for** $i \in \mathcal{I}_r$ **do**
10:        $s(i) \leftarrow$ **reconstruct**$(s, \rho, i, \mathscr{F})$
        {Algorithm 2}
11:        $w_{r,k(i)} \leftarrow \alpha(s(\rho(i)), s(i))$
12:     **end for**
13: **end for**
14: **process**$(s, w, h)$

---

**Algorithm 2** : **reconstruct**$(s, \rho, i, \mathscr{F} = \{F_i : i \in \mathcal{I}\})$

1: $F \leftarrow I$
2: **while** $(s(i) = \text{nil})$ **do**
3:     $F \leftarrow F \circ F_i$
4:     $i \leftarrow \rho(i)$
5: **end while**
6: **return** $F(s(i))$

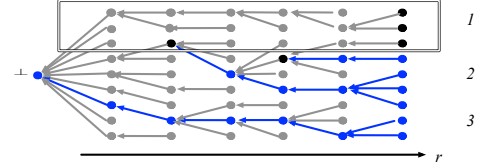

Figure 2: Illustration of compact particles (blue), concrete particles (black), and discarded particles (grey).

for only a subset $\mathcal{I}_r$ of the particles indices $\{1, \ldots, K\}$. We refer to machine $m$ as the *reference machine*. For brevity of notation, we omit notation $m$ when it is clear that we refer to the reference machine.

Parallelizing SMC is complicated by the resampling step. If roughly all particles were resampled exactly once, we would be able to assign to each machine the same indices as the previous iteration, avoiding communication. However, this rarely happens in practice. Instead, a small number of particles is often resampled a large number of times while many others have no offspring. This means that $\mathcal{I}_r$ can radically change across iterations. This raises an important question: how can a machine compute a proposal if the particle from which to propose was itself computed by a different machine?

The naive approach would consist in transmitting the 'missing' particles over the network. However, even if basic optimizations are used (for example sending particles with multiplicities only once), we show in Section 4 that this transfer can be slow in practice. Instead, our approach relies on a combination of the stochastic maps with the particle genealogy to *reconstruct* the particle. Let us see what this means in more detail, by going over the key steps of EMC, shown in Algorithm 1.

First, note that the resampling step in SMC algorithms induces a one-to-many relationship between the particle in generation $r$ and those in generation $r - 1$. This relationship is called the particle genealogy, illustrated in Figure 1 (b). Formally, a genealogy is a directed graph where nodes are particles $s_{r,k}$, $r \in \{1, \ldots, R\}$, $k \in \{1, \ldots, K\}$, and where node $s_{r-1,k}$ is deemed the *parent* of node $s_{r,k'}$ if the latter was obtained by resampling $\tilde{s}_{r-1,k'} = s_{r-1,k}$ followed by proposing $s_{r,k'}$ from $\tilde{s}_{r-1,k'}$.

Suppose for now that each machine kept track of the full genealogy, in the form of a hashtable $\rho : \mathcal{I} \to \mathcal{I}$ of parent pointers. Each machine also maintains a hashtable holding the particles held in memory in the reference machine $s : \mathcal{I} \to \mathcal{S} \cup \{\text{nil}\}$ (the value nil represent a particle not currently represented explicitly in the reference machine). Algorithm 2 shows that this information, $s, \rho, \mathscr{F}$, is sufficient to instantiate any query particle (indexed by $i$ in the pseudo-code). Note that the procedure **reconstruct** is guaranteed to terminate: in the procedure **init**, we set $s(i(0, k))$ to $\bot$, and the weights uniformly, hence $\bot$ is an ancestor of all particles.

This high-level description raises several questions. How can we efficiently store and retrieve the stochastic maps? Can we maintain a sparse view of the genealogical information $\rho, s$ to keep space requirements low? Finally, how can we do resampling and particle allocation in this distributed framework? We will cover these issues in the remaining of this section, describing at the same time how the procedures **allocate**, **resample** and **exchange** are implemented.

## 3.1 Allocation and resampling

In SMC algorithms, the weights are periodically used for resampling the particles, a step also known as the bootstrapping stage and denoted by **resample** in Algorithm 1. This is the only stage where EMC requires communication over the network to be done. With each machine having the full information of the weights in the current iteration, they can each perform a standard, global resampling step without further communication.

In most cases of interest, each machine can transmit all the individual weights of its particles and to communicate it with every other machine (either via a central server, or a decentralized scheme such as [13]) without becoming the bottleneck. Extreme cases, where even the list of weights alone is too large to transmit, can also be handled by transmitting only the sum of the weights of each machine, and using a distributed hashtable [13] to represent the genealogy. The modifications needed to implement this are discussed in Supplementary Material. We focus on the simpler case here.

Once the resampling step determines which particles survive to the next generation, the next step is to determine allocation of particles to machines. Particle allocation is an optimization problem where the objective is to reduce the reconstruction time with respect to the set of partition of particles.

Let $\{\mathcal{A}_r^1, \ldots, \mathcal{A}_r^M\}$ be the set of partition of particles $\{1, \ldots, K\}$ at generation $r$ and let $c_m$ denote the maximum number of particles that can be processed by machine $m$. For $i \in \mathcal{A}_r^m$, let $\Phi(i)$ be the number of times the stochastic map needs to be applied. The objective function is defined as,

$$\min_{\{\mathcal{A}_r^1, \ldots, A_r^M \text{ s.t } |\mathcal{A}_r^m| \leq c_m \forall m\}} \sum_{m=1}^{M} \sum_{i \in A_r^m} \Phi(i)$$

Obtaining an exact solution to this optimization problem is infeasible in practice as it requires enumerating over the set of all possible partitions. We propose greedy methods where each machine retains as many particles from $\mathcal{I}_{r-1}^m$ as possible. Let $\tilde{\mathcal{I}}_r^m \subseteq \mathcal{I}_{r-1}^m$ be the set of particles resampled from machine $m$. If $|\tilde{\mathcal{I}}_r^m| - c_m > 0$, this machine is in surplus of particles. We propose variety of greedy schemes to allocate the surplus of particles over to machines $m' \neq m$, where $c_{m'} - |\tilde{\mathcal{I}}_r^{m'}| > 0$.

**FirstOpen:** a deterministic scheme where a known list of preferred machines are known by all machines. The surplus particles are allocated according to this list.

**MostAvailable:** attempts to allocate the surplus particles to machines with the most capacity as defined by $c_{m'} - \tilde{\mathcal{I}}_r^{m'}$.

**Random:** samples a machine $m'$ at random with equal probability $1/M$. The intention is that the particles are mixed well over different machines so that the reconstruction algorithm rarely traces back the genealogy to the root ancestor.

## 3.2 Genealogy

In this section, we argue that for the purpose of reconstruction, only a sparse subset of the genealogy needs to be represented at any given iteration and machine. The key idea is that if a particle has no descendant in the current generation, storing its parent is not necessary. In practice, we observed that the vast majority of the ancestral particles have this property. We discuss at the end of this section some intuition as of why this holds, using a coalescent model.

Let us first look at how we can efficiently exploit this property. First, it is useful to draw a distinction between *concrete particles*, with $s(i) \neq$ nil, and *compact particles*, which are particles implicitly represented via an integer (the parent of the particle), and are therefore considerably more space-efficient. For example, in the smallest phylogenetic example considered in Section 4, a compact particle occupies about $50,000$ times less memory than a concrete particle. Whereas a concrete particle can grow in size as the problem size increases, a compact particle size is fixed.

Particles, concrete or compact, can become *obsolete*, meaning that the algorithm can guarantee that they will not be needed in subsequent iterations. This can happen for at least two reasons, each of which is efficiently detected at a different stage of the algorithm.

**Update after resampling**: Any lineage (path in $\rho$) that did not survive the resampling stage no longer needs to be maintained. This is illustrated in Figure 2. The greyed out particles will never

be reconstructed in the future generation so they are no longer maintained. Note that it is easy to harness a garbage collector to perform this update in practice.

**Update after reconstruction**: Once a particle is reconstructed, the lineage of the reconstructed particle can be updated. Let $j$ be the particle that is reconstructed at generation $r$. At any future generation $r' > r$, the reconstruction algorithm will only trace up to $j$ (as $s(j) \neq nil$), and hence all its parent can be discarded. Note that similar updates can be performed on $s$ to keep $s$ sparse as well.

The coalescent [14] can provide a potential theoretical model for understanding why these strategies are so effective in practice. If we assumed the weight function $\alpha$ to be constant, the genealogy induced by resampling can be viewed as a Wright-Fisher model [14, 15], which is well approximated by the coalescent when the number of particles is large. For example, this means that $(1-1/k)/(1-1/K)$ is the expected time spent waiting for the last $k$ copies to coalesce [15].

Note that the coalescent also gives an intuition for having Algorithm 2 terminating well before reaching the initial symbol $\perp$. Again, this reflects what we observed in our experiments.

### 3.3 Compact representation of the stochastic maps

The cardinality of the set of the stochastic maps $\mathscr{F} = \{F_i : i \in \mathcal{I}\}$ grows proportionally to the number of particles $K$ time the number of generations $R$. To store these maps naively would require the storage of $O(KR)$ uniform random variables $U_i$. However, since in practice pseudo-random numbers rather than true independent numbers are typically used, the sequence can be stored implicitly by maintaining only a random seed shared between machines. A drawback to this approach is that it is not efficient to perform random access of the random numbers. Random access of random numbers is an unusual requirement imposed by the genealogy reconstruction algorithm. Fortunately, as we discuss in this section, it is not hard to modify pseudo-random generators to support random access.

The simplest strategy to obtain faster random access is to cache intermediate internal states of the pseudo-random generator. For example by doing so for every particle generation, we get a faster access time of $O(K)$ and a larger space requirement of $O(R)$. More generally, this method can provide tradeoff of $O(n)$ space and $O(m)$ time with $mn = RK$.

In Supplementary Material, we describe the details of an alternative that requires $O(1)$ storage with $O(\log(KR))$ time for random access of any given map with index $i \in \mathcal{I}$. This method could potentially change the quality of the pseudo-random sequences obtained, but as described in Section 4.2, we have empirical evidence suggesting that the new pseudo-random scheme does not affect the quality of the estimated posterior expectations.

## 4 Experiments

In this section, we demonstrate the empirical performance of our method on synthetic and real datasets. As a first validation, we start by demonstrating that the behavior of our sampler equipped with our stochastic map datastructure is indistinguishable from that of a sampler based on a standard pseudo-random generator. Then we show results on the task of Bayesian phylogenetic inference, a challenging domain where massively parallel simulation is likely to have an impact for practitioners—running phylogenetic MCMC chains for months is not uncommon. To keep the exposition self-contained, we include a review of the phylogenetic SMC techniques we used.

### 4.1 Experimental setup

Given a collection of biological sequences for different species (taxa), Bayesian phylogenetics aims to compute expectations under a posterior distribution over *phylogenetic trees*, which represent the relationship among the species under study [16]. For intermediate to large numbers of species, Bayesian phylogenetic inference via SMC requires a large number of particles to achieve an accurate estimate. This is due to fact that the total number of distinct tree topologies increases at a super-exponential rate as the number of species increases [16].

In the following section, we use the phylogenetic SMC algorithm described in [2], where particles are proposed using a proposal with density $\nu(s \rightarrow s')$. Starting from a fully disconnected forest over the species, $\nu$ picks one pair of trees in the forest at random, and forms a new tree by connecting their roots. Under weak conditions described in [11], the following weight update yields a consistent estimator for the posterior over phylogenies:

$$\alpha(\tilde{s}_{r-1} \rightarrow s_r) \leftarrow \frac{\gamma(s_r)}{\gamma(\tilde{s}_{r-1})} \cdot \frac{1}{\nu(\tilde{s}_{r-1} \rightarrow s_r)},$$

where $\gamma$ is an unnormalized density over forests. In the experiments in Section 4.2 and Supplementary Material, where we wanted to run our SMC for more iterations, we use an alternation of kernel: in a first phase, the kernel described above, and in the second phase, the MCMC kernel of [17], transformed into a SMC kernel using the technique of [4, 5].

To generate synthetic datasets, we used a standard process [11]: we sampled trees from the coalescent, simulated data along the tree using a K2P likelihood model, discarded the values at internal nodes to keep only the observations at the leaves and held out the tree.

For real datasets, we used the manually aligned ribosomal RNA (rRNA) dataset of [18]. We used a subset of 28 sequences in the directory containing 5S rRNA sequences of Betaproteobacteria and a larger subset of 4,510 sequences of 16S rRNA sequences from Actinobacteria. We did experiments on two different numbers of subsampled species: 20 and 100.

## 4.2 Validation of the stochastic random maps datastructure

To check if the scheme described in Section 3.3 affects the quality of the SMC approximation of the target distribution, we carried out experiments to compare the quality of the SMC approximation based on pseudo-random numbers generated from **uniform** algorithm outlined in the Supplementary Material against the standard pseudo-random number generator. The dataset is a synthetic phylogenetics data with 20 taxa and 1000 sites. We measured a tree metric, the Robinson Foulds metric, on the consensus tree at every iteration, to detect potential biases in the estimator. We show random examples of pairs of runs in the Supplementary Material.

## 4.3 Speed-up results

In this section, we show experimental results where we measure the speed-up of an EMC algorithm on two sets of phylogenetics data by counting the number of times the maps $F_i$ are applied. The question we explore here is how deep the reconstruction algorithm has to trace back, or more precisely, how many times a parallelized version of our algorithm applies maps $F_i$ compared to the number of times the equivalent operation is performed in the serial version of SMC.

We denote $N_1$ to be the number of times the proposal function is applied in serial SMC, and $N_M$ to be the number of stochastic maps applied in our algorithm ran on $M$ machines. We measure the speedup as a ratio of $M$ and $R_M$, $S_M = \frac{M}{R_M}$ where $R_M = \frac{N_M}{N_1}$.

We ran these experiments on the 16S and 5S subsets of the rRNA data described earlier. In both subsets, we found a substantial speedup, suggesting that deep reconstruction was rarely needed in practice. We also obtained the following empirical ranking across the performance of the allocation methods: FirstOpen > MostAvailable > Random. We show the results on 100 taxa (species) for 5S and 16S in Figure 3.

We also performed an experiment on synthetic data generated with 20 taxa and 1000 sites that parallelization using EMC is beneficial in the corner case when the weights are all equal. For the purposes of illustration, we included an extra allocation method, *chaos*. This is an allocation method where particles are allocated at random, which disregards the greedy method suggested in Section 3.1. We show the results in Figure 3, where it can be seen that the speedup is still substantial in this context for all of the allocation methods.

## 4.4 Timing results

An SMC algorithm can easily be distributed over multiple machines by relying naively on particle transmission between machines over the network. In this section, we compared the particle transmission time to reconstruction time of EMC on Amazon EC2 micro instances.

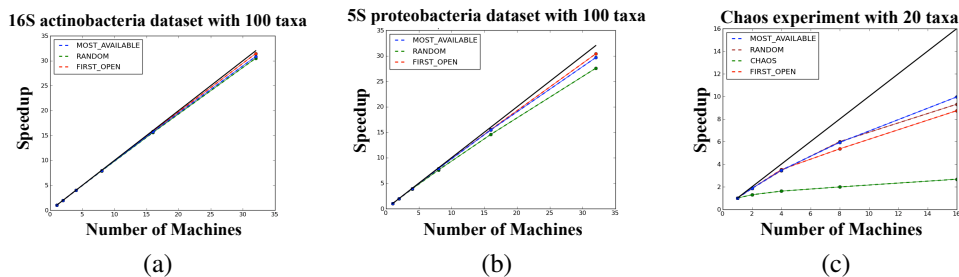

(a)           (b)           (c)

Figure 3: The speedup factor for (a) the 16S actinobacteria dataset with 100 taxa, (b) the 5S actinobacteria dataset with 100 taxa, and (c) the uniform weight synthetic experiment (see text).

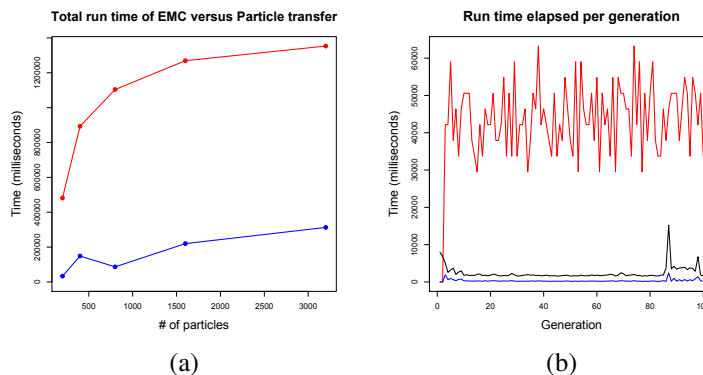

(a)           (b)

Figure 4: (a) Total time for particle transfer (red), total time for EMC (blue). (b) Sample generation time including reconstruction time (black), reconstruction time (blue), and particle transfer time (red) by generation.

The timing results in this section builds on the results from Section 4.3 where we showed that the ratio of $N_M$ and $N_1$ is small. Here, we ran SMC algorithm for 100 generations and measured the total run time of the EMC algorithm and an SMC algorithm parallelized via explicit particle transfer—see Figure 4 (a). We fixed the number of particles per machine at 100 and produced a sequence of experiments by doubling the number of machines and hence the number of particles at each step. In Figure 4 (b), we show the reconstruction time, the sample generation time (which includes the reconstruction time), and the particle transmission time by generation. As expected, the particle transmission is the bottleneck to the SMC algorithm whereas the reconstruction time is stable, which verifies that the reconstruction algorithm rarely traced deep.

The total timing result in Figure 4 (a) shows that the overhead arising from increasing the number of particles (or increasing the number of machines) is much smaller compared to the particle transmission method. The breakdown of time by generation in Figure 4 (b) shows that the particle transmission time is volatile as it depends on the network latency and throughput. The reconstruction time is stable as it relies only on the CPU cycles.

# 5 Conclusion

We have introduced EMC, a method to parallelize an SMC algorithm over multiple nodes. The new method requires only a small amount of data communication over the network, of size per particle independent of the scale of the inference problem. We have shown that the algorithm performs very well in practice on a Bayesian phylogenetic example and our software can be downloaded at `stat.ubc.ca/~seong.jun/`.

**Acknowledgements**

We thank Arnaud Doucet, Fabian Wauthier, and the anonymous reviewers for their helpful comments.

# References

[1] D. P. Anderson. BOINC: A System for Public-Resource Computing and Storage. In *GRID '04: Proceedings of the 5th IEEE/ACM International Workshop on Grid Computing*, pages 4–10, Washington, DC, USA, 2004. IEEE Computer Society.

[2] Y. W. Teh, H. Daumé III, and D. M. Roy. Bayesian agglomerative clustering with coalescents. In *Advances in Neural Information Processing Systems (NIPS)*, 2008.

[3] J. G. Propp and D. B. Wilson. Coupling from the past: a user's guide. *Microsurveys in Discrete Probability. DIMACS Series in Discrete Mathematics and Theoretical Computer Science*, 41:181–192, 1998.

[4] P. Del Moral, A. Doucet, and A. Jasra. Sequential Monte Carlo samplers. *Journal of The Royal Statistical Society Series B-statistical Methodology*, 68(3):411–436, 2006.

[5] P. Del Moral, A. Doucet, and A. Jasra. Sequential Monte Carlo for Bayesian computation. *Bayesian Statistics*, 8:1–34, 2007.

[6] A. Lee, C. Yau, M. B. Giles, A. Doucet, and C. C. Holmes. On the utility of graphics cards to perform massively parallel simulation of advanced Monte Carlo methods. *Journal of Computational and Graphical Statistics*, 19(4):769–789, 2010.

[7] S. Singh and A. McCallum. Towards asynchronous distributed MCMC inference for large graphical models. In *Neural Information Processing Systems (NIPS), Big Learning Workshop on Algorithms, Systems, and Tools for Learning at Scale*, 2011.

[8] J. Gonzalez, Y. Low, A. Gretton, and C. Guestrin. Parallel Gibbs sampling: From colored fields to thin junction trees. In *In Artificial Intelligence and Statistics (AISTATS)*, Ft. Lauderdale, FL, May 2011.

[9] S. Singh, A. Subramanya, F. Pereira, and A. McCallum. Large-scale cross-document coreference using distributed inference and hierarchical models. In *Association for Computational Linguistics: Human Language Technologies (ACL HLT)*, 2011.

[10] R. H. Swendsen and J-S. Wang. Replica Monte Carlo simulation of spin-glasses. *Phys. Rev. Lett.*, 57:2607–2609, Nov 1986.

[11] A. Bouchard-Côté, S. Sankararaman, and M. I. Jordan. Phylogenetic inference via Sequential Monte Carlo. *Systematic Biology*, 2011.

[12] A. Doucet, N. de Freitas, and N. Gordon. *Sequential Monte Carlo methods in practice*. Springer, 2001.

[13] I. Stoica, R. Morris, D. Karger, M. F. Kaashoek, and H. Balakrishnan. Chord: A scalable peer-to-peer lookup service for internet applications. *ACM SIGCOMM 2001*, pages 149–160, 2001.

[14] J. F. C. Kingman. On the Genealogy of Large Populations. *Journal of Applied Probability*, 19:27–43, 1982.

[15] J. Felsenstein. *Inferring phylogenies*. Sinauer Associates, 2003.

[16] C. Semple and M. Steel. *Phylogenetics*. Oxford, 2003.

[17] J. P. Huelsenbeck and F. Ronquist. MRBAYES: Bayesian inference of phylogenetic trees. *Bioinformatics*, 17(8):754–755, August 2001.

[18] J.J. Cannone, S. Subramanian, M.N. Schnare, J.R. Collett, L.M. D'Souza, Y. Du, B. Feng, N. Lin, L.V. Madabusi, K.M. Muller, N. Pande, Z. Shang, N. Yu, and R.R. Gutell. The comparative RNA web (CRW) site: An online database of comparative sequence and structure information for ribosomal, intron, and other RNAs. *BioMed Central Bioinformatics*, 2002.

